# Gradient-based kernel method for feature extraction and variable selection

**Kenji Fukumizu**
The Institute of Statistical Mathematics
10-3 Midori-cho, Tachikawa, Tokyo 190-8562 Japan
fukumizu@ism.ac.jp

**Chenlei Leng**
National University of Singapore
6 Science Drive 2, Singapore, 117546
stalc@nus.edu.sg

## Abstract

We propose a novel kernel approach to dimension reduction for supervised learning: feature extraction and variable selection; the former constructs a small number of features from predictors, and the latter finds a subset of predictors. First, a method of linear feature extraction is proposed using the gradient of regression function, based on the recent development of the kernel method. In comparison with other existing methods, the proposed one has wide applicability without strong assumptions on the regressor or type of variables, and uses computationally simple eigendecomposition, thus applicable to large data sets. Second, in combination of a sparse penalty, the method is extended to variable selection, following the approach by Chen et al. [2]. Experimental results show that the proposed methods successfully find effective features and variables without parametric models.

## 1 Introduction

Dimension reduction is involved in most of modern data analysis, in which high dimensional data must be handled. There are two categories of dimension reduction: *feature extraction*, in which a linear or nonlinear mapping to a low-dimensional space is pursued, and *variable selection*, in which a subset of variables is selected. This paper discusses both the methods in supervised learning.

Let $(X, Y)$ be a random vector such that $X = (X^1, \ldots, X^m) \in \mathbb{R}^m$. The domain of $Y$ can be arbitrary, either continuous, discrete, or structured. The goal of dimension reduction in supervised setting is to find such features or a subset of variables $X$ that explain $Y$ as effectively as possible. This paper focuses linear dimension reduction, in which linear combinations of the components of $X$ are used to make effective features. Although there are many methods for extracting nonlinear features, this paper confines its attentions on linear features, since linear methods are more stable than nonlinear feature extraction, which depends strongly on the choice of the nonlinearity, and after establishing a linear method, extension to a nonlinear one would not be difficult.

We first develop a method for linear feature extraction with kernels, and extend it to variable selection with a sparseness penalty. The most significant point of the proposed methods is that we do not assume any parametric models on the conditional probability, or make strong assumptions on the distribution of variables. This differs from many other methods, particularly for variable selection, where a specific parametric model is often assumed. Beyond the classical approaches such as Fisher Discriminant Analysis and Canonical Correlation Analysis to linear dimension reduction, the modern approach is based on the notion of conditional independence; we assume for the distribution

$$p(Y|X) = \tilde{p}(Y|B^T X) \qquad \text{or equivalently} \qquad Y \perp\!\!\!\perp X \mid B^T X, \qquad (1)$$

where $B$ is a projection matrix ($B^T B = I_d$) onto a $d$-dimensional subspace ($d < m$) in $\mathbb{R}^m$, and wish to estimate $B$. For variable selection, we further assume that some rows of $B$ may be zero. The subspace spanned by the columns of $B$ is called the *effective direction for regression*, or *EDR space* [14]. Our goal is thus to estimate $B$ without specific parametric models for $p(y|x)$.

First, consider the linear feature extraction based on Eq. (1). The first method using this formulation is the *sliced inverse regression* (SIR, [13]), which employs the fact that the inverse regression $E[X|Y]$ lies in the EDR space under some assumptions. Many methods have been proposed in this vein of inverse regression ([4, 12] among others). While the methods are computationally simple, they often need some strong assumptions on the distribution of $X$ such as elliptic symmetry.

There are two most relevant works to this paper. The first one is the dimension reduction with the gradient of regressor $E[Y|X = x]$ [11, 17]. As explained in Sec. 2.1, under Eq. (1) the gradient is contained in the EDR space. One can thus estimate the space by some standard nonparametric method. There are some limitations in this approach, however: the nonparametric gradient estimation in high-dimensional spaces is challenging, and the method may not work unless the noise is additive. The second one is the kernel dimension reduction (KDR, [8, 9, 28]), which uses the kernel method for characterizing the conditional independence to overcome various limitations of existing methods. While KDR applies to a wide class of problems without any strong assumptions on the distributions or types of $X$ or $Y$, and shows high estimation accuracy for small data sets, its optimization has a problem: the gradient descent method used for KDR may have local optima, and needs many matrix inversions, which prohibits application to high-dimensional or large data.

We propose a kernel method for linear feature extraction using the gradient-based approach, but unlike the existing ones [11, 17], the gradient is estimated based on the recent development of the kernel method [9, 19]. It solves the problems of existing methods: by virtue of the kernel method, $Y$ can be of arbitrary type, and the kernel estimator is stable without careful decrease of bandwidth. It solves also the problem of KDR: the estimator by an eigenproblem needs no numerical optimization. The method is thus applicable to large and high-dimensional data, as we demonstrate experimentally.

Second, by using the above feature extraction in conjunction with a sparseness penalty, we propose a novel method for variable selection. Recently extensive studies have been done for variable selection with a sparseness penalty such as LASSO [23] and SCAD [6]. It is also known that with appropriate choice of regularization coefficients they have oracle property [6, 25, 30]. These methods, however, use some specific model for regression such as linear regression, which is a limitation of the methods. Chen et al. [2] proposed a novel method for sparse variable selection based on the objective function of linear feature extraction formulated as an eigenproblem such as SIR. We follow this approach to derive our method for variable selection. Unlike the methods used in [2], the proposed one does not require strong assumptions on the regressor or distribution, and thus provides a variable selection method based on the conditional independence irrespective of the regression model.

## 2 Gradient-based kernel dimension reduction

### 2.1 Gradient of a regression function and dimension reduction

We review the basic idea of the gradient-based method [11, 17] for dimension reduction. Suppose $Y$ is an $\mathbb{R}$-valued random variable. If the assumption of Eq. (1) holds, we have

$$\frac{\partial}{\partial x}E[Y|X = x] = \frac{\partial}{\partial x}\int yp(y|x)dy = \int y\frac{\partial}{\partial x}\tilde{p}(y|B^T x)dy = B\int y \left.\frac{\partial}{\partial z}\tilde{p}(y|z)\right|_{z=B^T x} dy,$$

which implies that the gradient $\frac{\partial}{\partial x}E[Y|X = x]$ at any $x$ is contained in the EDR space. Based on this fact, the average derivative estimates (ADE, [17]) has been proposed to estimate $B$. In the more recent method [11], a standard local linear least squares with a smoothing kernel (not necessarily positive definite, [5]) is used for estimating the gradient, and the dimensionality of the projection is continuously reduced to the desired one in the iteration. Since the gradient estimation for high-dimensional data is difficult in general, the iterative reduction is expected to give more accurate estimation. We call the method in [11] iterative average derivative estimates (IADE) in the sequel.

### 2.2 Kernel method for estimating gradient of regression

For a set $\Omega$, a ($\mathbb{R}$-valued) *positive definite kernel* $k$ on $\Omega$ is a symmetric kernel $k : \Omega \times \Omega \to \mathbb{R}$ such that $\sum_{i,j=1}^{n} c_i c_j k(x_i, x_j) \geq 0$ for any $x_1, \ldots, x_n$ in $\Omega$ and $c_1, \ldots, c_n \in \mathbb{R}$. It is known that a positive definite kernel on $\Omega$ uniquely defines a Hilbert space $\mathcal{H}$ consisting of functions on $\Omega$, in which the reproducing property $\langle f, k(\cdot, x)\rangle_{\mathcal{H}} = f(x)$ ($\forall f \in \mathcal{H}$) holds, where $\langle \cdot, \cdot \rangle_{\mathcal{H}}$ is the inner product of $\mathcal{H}$. The Hilbert space $\mathcal{H}$ is called the *reproducing kernel Hilbert space* (RKHS) associated with $k$. We assume that an RKHS is always separable.

In deriving a kernel method based on the approach in Sec. 2.1, the fundamental tool is the reproducing property for the derivative of a function. It is known (*e.g.*, [21] Sec. 4.3) that if a positive definite kernel $k(x,y)$ on an open set in the Euclidean space is continuously differentiable with respect to $x$ and $y$, every $f$ in the corresponding RKHS $\mathcal{H}$ is continuously differentiable. If further $\frac{\partial}{\partial x}k(\cdot,x) \in \mathcal{H}$, we have

$$\frac{\partial f}{\partial x} = \left\langle f, \frac{\partial}{\partial x}k(\cdot,x)\right\rangle_{\mathcal{H}}. \tag{2}$$

This reproducing property combined with the following kernel estimator of the conditional expectation (see [8, 9, 19] for details) will provide a method for dimension reduction. Let $(X,Y)$ be a random variable on $\mathcal{X} \times \mathcal{Y}$ with probability $P$. We always assume that the p.d.f. $p(x,y)$ and the conditional p.d.f. $p(y|x)$ exist, and that a positive definite kernel is measurable and bounded. Let $k_{\mathcal{X}}$ and $k_{\mathcal{Y}}$ be positive definite kernels on $\mathcal{X}$ and $\mathcal{Y}$, respectively, with respective RKHS $\mathcal{H}_{\mathcal{X}}$ and $\mathcal{H}_{\mathcal{Y}}$. The (uncentered) *covariance operator* $C_{YX}: \mathcal{H}_{\mathcal{X}} \to \mathcal{H}_{\mathcal{Y}}$ is defined by the equation

$$\langle g, C_{YX}f\rangle_{\mathcal{H}_{\mathcal{Y}}} = E[f(X)g(Y)] = E\big[\langle f, \Phi_{\mathcal{X}}(X)\rangle_{\mathcal{H}_{\mathcal{X}}}\langle\Phi_{\mathcal{Y}}(Y),g\rangle_{\mathcal{H}_{\mathcal{Y}}}\big] \tag{3}$$

for all $f \in \mathcal{H}_{\mathcal{X}}, g \in \mathcal{H}_{\mathcal{Y}}$, where $\Phi_{\mathcal{X}}(x) = k_{\mathcal{X}}(\cdot,x)$ and $\Phi_{\mathcal{Y}}(y) = k_{\mathcal{Y}}(\cdot,y)$. Similarly, $C_{XX}$ denotes the operator on $\mathcal{H}_{\mathcal{X}}$ that satisfies $\langle f_2, C_{XX}f_1\rangle = E[f_2(X)f_1(X)]$ for any $f_1, f_2 \in \mathcal{H}_{\mathcal{X}}$. These definitions are straightforward extensions of the ordinary covariance matrices, if we consider the covariance of the random vectors $\Phi_{\mathcal{X}}(X)$ and $\Phi_{\mathcal{Y}}(Y)$ on the RKHSs. One of the advantages of the kernel method is that estimation with finite data is straightforward. Given i.i.d. sample $(X_1,Y_1),\ldots,(X_n,Y_n)$ with law $P$, the covariance operator is estimated by

$$\widehat{C}_{YX}^{(n)}f = \tfrac{1}{n}\sum_{i=1}^{n}k_{\mathcal{Y}}(\cdot,Y_i)\langle k_{\mathcal{X}}(\cdot,X_i),f\rangle_{\mathcal{H}_{\mathcal{X}}} \quad \widehat{C}_{XX}^{(n)}f = \tfrac{1}{n}\sum_{i=1}^{n}k_{\mathcal{X}}(\cdot,X_i)\langle k_{\mathcal{X}}(\cdot,X_i),f\rangle_{\mathcal{H}_{\mathcal{X}}}. \tag{4}$$

It is known [8] that if $E[g(Y)|X=\cdot] \in \mathcal{H}_{\mathcal{X}}$ holds for $g \in \mathcal{H}_{\mathcal{Y}}$, then we have $C_{XX}E[g(Y)|X=\cdot] = C_{XY}g$. If further $C_{XX}$ is injective[1], this relation can be expressed as

$$E[g(Y)|X=\cdot] = C_{XX}^{-1}C_{XY}g. \tag{5}$$

While the assumption $E[g(Y)|X=\cdot] \in \mathcal{H}_{\mathcal{X}}$ may not hold in general, we can nonetheless obtain an empirical estimator based on Eq. (5), namely,

$$(\widehat{C}_{XX}^{(n)} + \varepsilon_n I)^{-1}\widehat{C}_{XY}^{(n)}g,$$

where $\varepsilon_n$ is a regularization coefficient in Tikhonov-type regularization. Note that the above expression is the kernel ridge regression of $g(Y)$ on $X$. As we discuss in Supplements, we can in fact prove rigorously that this estimator converges to $E[g(Y)|X=\cdot]$.

Assume now that $\mathcal{X} = \mathbb{R}^m$, $C_{XX}$ is injective, $k_{\mathcal{X}}(x,\tilde{x})$ is continuously differentiable, $E[g(Y)|X=x] \in \mathcal{H}_{\mathcal{X}}$ for any $g \in \mathcal{H}_{\mathcal{Y}}$, and $\frac{\partial}{\partial x}k_{\mathcal{X}}(\cdot,x) \in \mathcal{R}(C_{XX})$, where $\mathcal{R}$ denotes the range of the operator. From Eqs. (5) and (2), $\frac{\partial}{\partial x}E[g(Y)|X=x] = \langle C_{XX}^{-1}C_{XY}g, \frac{\partial k_{\mathcal{X}}(\cdot,x)}{\partial x}\rangle = \langle g, C_{YX}C_{XX}^{-1}\frac{\partial k_{\mathcal{X}}(\cdot,x)}{\partial x}\rangle$. With $g = k_{\mathcal{Y}}(\cdot,\tilde{y})$, we obtain the gradient of regression of the feature vector $\Phi_{\mathcal{Y}}(Y)$ on $X$ as

$$\frac{\partial}{\partial x}E[\Phi_{\mathcal{Y}}(Y)|X=x] = C_{YX}C_{XX}^{-1}\frac{\partial k_{\mathcal{X}}(\cdot,x)}{\partial x}. \tag{6}$$

## 2.3 Gradient-based kernel method for linear feature extraction

It follows from the same argument as in Sec. 2.1 that $\frac{\partial}{\partial x}E[k_{\mathcal{Y}}(\cdot,y)|X=x] = \Xi(x)B$ with an operator $\Xi(x)$ from $\mathbb{R}^m$ to $\mathcal{H}_{\mathcal{Y}}$, where we use a slight abuse of notation by identifying the operator $\Xi(x)$ with a matrix. In combination with Eq. (6), we have

$$B^T\langle\Xi(x),\Xi(x)\rangle_{\mathcal{H}_{\mathcal{Y}}}B = \left\langle\frac{\partial k_{\mathcal{X}}(\cdot,x)}{\partial x}, C_{XX}^{-1}C_{XY}C_{YX}C_{XX}^{-1}\frac{\partial k_{\mathcal{X}}(\cdot,x)}{\partial x}\right\rangle_{\mathcal{H}_{\mathcal{X}}} =: M(x), \tag{7}$$

which shows that the eigenvectors for non-zero eigenvalues of $m \times m$ matrix $M(x)$ are contained in the EDR space. This fact is the basis of our method. In contrast to the conventional gradient-based method described in Sec. 2.1, this method incorporates high (or infinite) dimensional regressor $E[\Phi_{\mathcal{Y}}(Y)|X=x]$.

Given i.i.d. sample $(X_1, Y_1), \ldots, (X_n, Y_n)$ from the true distribution, based on the empirical covariance operators Eq. (4) and regularized inversions, the matrix $M(x)$ is estimated by

$$\widehat{M}_n(x) = \left\langle \frac{\partial k_{\mathcal{X}}(\cdot, x)}{\partial x}, \left(\widehat{C}_{XX}^{(n)} + \varepsilon_n I\right)^{-1} \widehat{C}_{XY}^{(n)} \widehat{C}_{YX}^{(n)} \left(\widehat{C}_{XX}^{(n)} + \varepsilon_n I\right)^{-1} \frac{\partial k_{\mathcal{X}}(\cdot, x)}{\partial x} \right\rangle$$

$$= \nabla \mathbf{k}_X(x)^T (G_X + n\varepsilon_n I)^{-1} G_Y (G_X + n\varepsilon_n I)^{-1} \nabla \mathbf{k}_X(x), \tag{8}$$

where $G_X$ and $G_Y$ are the Gram matrices $(k_{\mathcal{X}}(X_i, X_j))$ and $(k_{\mathcal{Y}}(Y_i, Y_j))$, respectively, and $\nabla \mathbf{k}_X(x) = (\partial k_{\mathcal{X}}(X_1, x)/\partial x, \cdots, \partial k_{\mathcal{X}}(X_n, x)/\partial x)^T \in \mathbb{R}^n$.

As the eigenvectors of $M(x)$ are contained in the EDR space for any $x$, we propose to use the average of $M(X_i)$ over all the data points $X_i$, and define

$$\tilde{M}_n := \frac{1}{n} \sum_{i=1}^n \widehat{M}_n(X_i) = \frac{1}{n} \sum_{i=1}^n \nabla \mathbf{k}_X(X_i)^T (G_X + n\varepsilon_n I_n)^{-1} G_Y (G_X + n\varepsilon_n I_n)^{-1} \nabla \mathbf{k}_X(X_i).$$

We call the dimension reduction with the matrix $\tilde{M}_n$ the *gradient-based kernel dimension reduction* (gKDR). For linear feature extraction, the projection matrix $B$ in Eq. (1) is then estimated simply by the top $d$ eigenvectors of $\tilde{M}_n$. We call this method gKDR-FEX.

The proposed method applies to a wide class of problems; in contrast to many existing methods, the gKDR-FEX can handle any type of data for $Y$ including multinomial or structured variables, and make no strong assumptions on the regressor or distribution of $X$. Additionally, since the gKDR incorporates the high dimensional feature vector $\Phi_{\mathcal{Y}}(Y)$, it works for any regression relation including multiplicative noise, for which many existing methods such as SIR and IADE fail.

As in all kernel methods, the results of gKDR depend on the choice of kernels. We use the cross-validation (CV) for choosing kernels and parameters, combined with some regression or classification method. In this paper, the k-nearest neighbor (kNN) regression / classification is used in CV for its simplicity: for each candidate of a kernel or parameter, we compute the CV error by the kNN method with $(B^T X_i, Y_i)$, where $B$ is given by gKDR, and choose the one that gives the least error.

The time complexity of the matrix inversions and the eigendecomposition for gKDR are $O(n^3)$, which is prohibitive for large data sets. We can apply, however, low-rank approximation of Gram matrices, such as incomplete Cholesky decomposition. The space complexity may be also a problem of gKDR, since $(\nabla \mathbf{k}_X(X_i))_{i=1}^n$ has $n^2 \times m$ dimension. In the case of Gaussian kernel, where $\frac{\partial}{\partial x^a} k_X(X_j, x)|_{x=X_i} = \frac{1}{\sigma^2}(X_j^a - X_i^a) \exp(-\|X_j - X_i\|^2/(2\sigma^2))$, we have a way of reducing the necessary memory by low rank approximation. Let $G_X \approx RR^T$ and $G_Y \approx HH^T$ be the low rank approximation with $r_x = \mathrm{rk}R, r_y = \mathrm{rk}H$ ($r_x, r_y < n, m$). With the notation $F := (G_X + n\varepsilon_n I_n)^{-1} H$ and $\Theta_i^{as} = \frac{1}{\sigma^2} X_i^a R_{is}$, we have, for $1 \le a, b \le m$,

$$\tilde{M}_{n,ab} = \sum_{i=1}^n \sum_{t=1}^{r_y} \Gamma_{ia}^t \Gamma_{ib}^t, \quad \Gamma_{ia}^t = \sum_{s=1}^{r_x} R_{is}\left(\sum_{j=1}^n \Theta_j^{as} F_{jt}\right) - \sum_{s=1}^{r_x} \Theta_i^{as}\left(\sum_{j=1}^n R_{js} F_{jt}\right).$$

With this method, the complexity is $O(nmr)$ in space and $O(nm^2 r)$ in time ($r = \max\{r_x, r_y\}$), which is much more efficient in memory than straightforward implementation.

We introduce two variants of gKDR-FEX. First, since accurate nonparametric estimation with high-dimensional $X$ is not easy, we propose a method for decreasing the dimensionality iteratively. Using gKDR-FEX, we first find a matrix $B_1$ of dimensionality $d_1$ larger than the target $d$, project data $X_i$ onto the subspace as $Z_i^{(1)} = B_1^T X_i$, find the projection matrix $B_2$ ($d_1 \times d_2$ matrix) for $Z_i^{(1)}$ onto a $d_2$ ($d_2 < d_1$) dimensional subspace, and repeat this process. We call this method gKDR-FEXi.

Second, if $Y$ takes only $L$ points as in classification, the Gram matrix $G_Y$ and thus $\tilde{M}_n$ are of rank $L$ at most (see Eq. (8)), which is a strong limitation of gKDR. Note that this problem is shared by many linear dimension reduction methods including CCA and slice-based methods. To solve this problem, we propose to use the variation of $\widehat{M}_n(x)$ over the points $x = X_i$ instead of the average $\tilde{M}_n$. By partitioning $\{1, \ldots, n\}$ into $T_1, \ldots, T_\ell$, the projection matrices $\widehat{B}_{[a]}$ given by the eigenvectors of $\widehat{M}_{[a]} = \sum_{i \in T_a} \widehat{M}(X_i)$ are used to define $\widehat{P} = \frac{1}{\ell} \sum_{a=1}^\ell \widehat{B}_{[a]} \widehat{B}_{[a]}^T$. The estimator of $B$ is then given by the top $d$ eigenvectors of $\widehat{P}$. We call this method gKDR-FEXv.

## 2.4 Theoretical analysis of gKDR

We have derived the gKDR method based on the necessary condition of EDR space. The following theorem shows that it is also sufficient, if $k_{\mathcal{Y}}$ is characteristic. A positive definite kernel $k$ on a

| | gKDR-FEX | gKDR-FEXi | gKDR-FEXv | IADE | SIR II | KDR | gKDR-FEX+KDR |
|---|---|---|---|---|---|---|---|
| (A) $n=100$ | 0.1989 | 0.1639 | 0.2002 | 0.1372 | 0.2986 | 0.2807 | 0.0883 |
| (A) $n=200$ | 0.1264 | 0.0995 | 0.1287 | 0.0857 | 0.2077 | 0.1175 | 0.0501 |
| (B) $n=100$ | 0.1500 | 0.1358 | 0.1630 | 0.1690 | 0.3137 | 0.2138 | 0.1076 |
| (B) $n=200$ | 0.0755 | 0.0750 | 0.0802 | 0.0940 | 0.2129 | 0.1440 | 0.0506 |
| (C) $n=200$ | 0.1919 | 0.2322 | 0.1930 | 0.7724 | 0.7326 | 0.1479 | 0.1285 |
| (C) $n=400$ | 0.1346 | 0.1372 | 0.1369 | 0.7863 | 0.7167 | 0.0897 | 0.0893 |

Table 1: gKDE-FEX for synthetic data: mean discrepancies over 100 runs.

measurable space is *characteristic* if $E_P[k(\cdot, X)] = E_Q[k(\cdot, X)]$ means $P = Q$, i.e., the mean of feature vector uniquely determines a probability [9, 20]. Examples include Gaussian kernel.

In the following theoretical results, we assume (i) $\partial k_{\mathcal{X}}(\cdot, x)/\partial x^a \in \mathcal{R}(C_{XX})$ $(a = 1, \ldots, m)$, (ii) $E[k_{\mathcal{Y}}(y, X)|X = \cdot] \in \mathcal{H}_{\mathcal{X}}$ for any $y \in \mathcal{Y}$, and (iii) $E[g(Y)|B^T X = z]$ is a differentiable function of $z$ for any $g \in \mathcal{H}_{\mathcal{Y}}$ and the linear functional $g \mapsto \partial E[g(Y)|B^T X = z]/\partial z$ is continuous for any $z$. In the sequel, the subspace spanned by the columns of $B$ is denoted by $\mathrm{Span}(B)$, and the Frobenius norm of a matrix $M$ by $\|M\|_F$. The proofs are given in Supplements.

**Theorem 1.** *In addition to the above assumptions (i)-(iii), assume that the kernel $k_{\mathcal{Y}}$ is characteristic. If the eigenspaces for the non-zero eigenvalues of $E[M(X)]$ are included in $\mathrm{Span}(B)$, then $Y$ and $X$ are conditionally independent given $B^T X$.*

We can obtain the rate of consistency for $\widehat{M}_n(x)$ and $\tilde{M}_n$.

**Theorem 2.** *In addition to (i)-(iii), assume that $\frac{\partial k_{\mathcal{X}}(\cdot, x)}{\partial x^a} \in \mathcal{R}(C_{XX}^{\beta+1})$ $(a = 1, \ldots, m)$ for some $\beta \geq 0$, and $E[k_{\mathcal{Y}}(y, Y)|X = \cdot] \in \mathcal{H}_{\mathcal{X}}$ for every $y \in \mathcal{Y}$. Then, for $\varepsilon_n = n^{-\max\{\frac{1}{3}, \frac{1}{2\beta+2}\}}$, we have*

$$\widehat{M}_n(x) - M(x) = O_p\big(n^{-\min\{\frac{1}{3}, \frac{2\beta+1}{4\beta+4}\}}\big)$$

*for every $x \in \mathcal{X}$ as $n \to \infty$. If further $E[\|M(X)\|_F^2] < \infty$ and $\frac{\partial k_{\mathcal{X}}(\cdot, x)}{\partial x^a} = C_{XX}^{\beta+1} h_x^a$ with $E\|h_X^a\|_{\mathcal{H}_{\mathcal{X}}} < \infty$, then $\tilde{M}_n \to E[M(X)]$ in the same order as above.*

Note that, assuming that the eigenvalues of $M(x)$ or $E[M(X)]$ are all distinct, the convergence of matrices implies the convergence of the eigenvectors [22], thus the estimator of gKDR-FEX is consistent to the subspace given by the top eigenvectors of $E[M(X)]$.

## 2.5 Experiments with gKDR-FEX

We always use the Gaussian kernel $k(x, \tilde{x}) = \exp(-\frac{1}{2\sigma^2}\|x - \tilde{x}\|^2)$ in the kernel method below. First we use three synthetic data to verify the basic performance of gKDR-FEX(i,v). The data are generated by (A): $Y = Z \sin(\sqrt{5}Z) + W$, $Z = \frac{1}{\sqrt{5}}(1, 2, 0, \ldots, 0)^T X$, (B): $Y = (Z_1^3 + Z_2)(Z_1 - Z_2^3) + W$, $Z_1 = \frac{1}{\sqrt{2}}(1, 1, 0, \ldots, 0)^T X$, $Z_2 = \frac{1}{\sqrt{2}}(1, -1, 0, \ldots, 0)^T X$, where 10-dimensional $X$ is generated by the uniform distribution on $[-1, 1]^{10}$ and $W$ is independent noise with $N(0, 10^{-2})$, and (C): $Y = Z^4 E$, $Z = (1, 0, \ldots, 0)^T X$, where each component of 10-dimensional $X$ is independently generated by the truncated normal distribution $N(0, 1/4) * I_{[-1,1]}$ and $E \sim N(0, 1)$ is a multiplicative noise. The discrepancy between the estimator $B$ and the true projector $B_0$ is measured by $\|B_0 B_0^T (I_m - BB^T)\|_F/d$. For choosing the parameter $\sigma$ in Gaussian kernel and the regularization parameter $\varepsilon_n$, the CV in Sec. 2.3 with kNN (k = 5, manually chosen to optimize the results) is used with 8 different values given by $c\sigma_{med}$ ($0.5 \leq c \leq 10$), where $\sigma_{med}$ is the median of pairwise distances of data [10], and $\ell = 4, 5, 6, 7$ for $\varepsilon_n = 10^{-\ell}$ (a similar strategy is used for the CV below).

We compare the results with those of IADE, SIR II [13], and KDR. The IADE has seven parameters [11], and we tuned two of them ($h_1$ and $\rho_{\min}$) manually to optimize the performance. For SIR II, we tried several numbers of slices, and chose the one that gave the best result. From Table 1, we see that gKDR-FEX(i,v) show much better results than SIR II in all the cases. The IADE works better than these methods for (A), while for (B) and (C) it works worse. Since (C) has multiplicative noise, the IADE does not obtain meaningful estimation. The KDR attains higher accuracy for (C), but less accurate for (A) and (B) with $n = 100$; this undesired result is caused by failure of optimization in

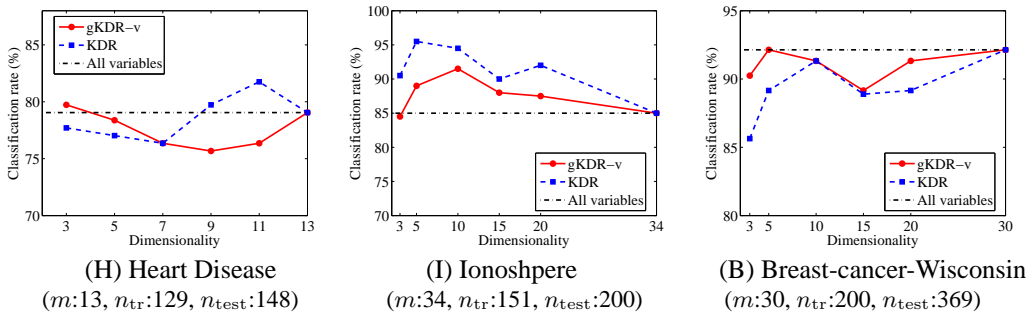

|   | (H) Heart Disease | (I) Ionoshpere | (B) Breast-cancer-Wisconsin |
|---|---|---|---|
|   | $(m{:}13, n_{\text{tr}}{:}129, n_{\text{test}}{:}148)$ | $(m{:}34, n_{\text{tr}}{:}151, n_{\text{test}}{:}200)$ | $(m{:}30, n_{\text{tr}}{:}200, n_{\text{test}}{:}369)$ |

Figure 1: Classification accuracy with gKDR-v and KDR for binary classification problems. $m$, $n_{\text{tr}}$ and $n_{\text{test}}$ are the dimension of $X$, training data size, and testing data size, respectively.

| Dim. | 10 | 20 | 30 | 40 | 50 |
|---|---|---|---|---|---|
| gKDR + kNN | 13.53 | 4.55 | – | – | – |
| gKDR-v + kNN | 13.15 | 4.55 | 4.81 | 5.26 | 5.58 |
| CCA + kNN | 22.77 | 6.74 | – | – | – |
| SIR-II + kNN | 77.42 | 70.11 | 63.44 | 52.66 | 50.61 |
| gKDR + SVM | 14.43 | 5.00 | – | – | – |
| gKDR-v + SVM | 16.87 | 4.75 | 3.85 | 3.59 | 3.08 |
| CCA + SVM | 13.09 | 6.54 | – | – | – |

| $L$ | gKDR +SVM | Corr +SVM (500) | Corr +SVM (2000) |
|---|---|---|---|
| 10 | 12.0 | 15.7 | 8.3 |
| 20 | 16.2 | 30.2 | 18.0 |
| 30 | 18.0 | 29.2 | 24.0 |
| 40 | 21.8 | 35.4 | 25.0 |
| 50 | 19.5 | 41.1 | 29.0 |

Table 2: Left: ISOLET - classification errors for test data (percentage). Right: Amazon Reviews - 10-fold cross-validation errors (%) for classification

some runs (see Supplements for error bars). We also used the results of gKDR-FEX as the initial state for KDR, which improved the accuracy significantly for (A) and (B). Note however that these data sets are very small in size and dimension, and KDR is not applicable to large data used later.

One way of evaluating dimension reduction methods in supervised learning is to consider the classification or regression accuracy after projecting data onto the estimated subspaces. We next used three data sets for binary classification, *heart-disease* (H), *ionoshpere* (I), and *breast-cancer-Wisconsin* (B), from UCI repository [7], and evaluated the classification rates of gKDR-FEXv with kNN classifiers (k = 7). We compared them with KDR, as KDR shows high accuracy for small data sets. From Fig. 1, we see gKDR-FEXv shows competitive accuracy with KDR: slightly worse for (I), and slightly better for (B). The computation of gKDR-FEXv for these data sets can be much faster than that of KDR. For each parameter set, the computational time of gKDR vs KDR was, in (H) 0.044 sec / 622 sec ($d = 11$), in (I) 0.l03 sec / 84.77 sec ($d = 20$), and in (B) 0.116 sec / 615 sec ($d = 20$).

The next two data sets taken from UCI repository are larger in the sample size and dimensionality, for which the optimization of KDR is difficult to apply. The first one is ISOLET, which provides 617 dimensional continuous features of speech signals to classify 26 alphabets. In addition to 6238 training data, 1559 test data are separately provided. We evaluate the classification errors with the kNN classifier (k = 5) and 1-vs-1 SVM to see the effectiveness of the estimated subspaces (see Table 2). From the information on the data at the UCI repository, the best performance with neural networks and C4.5 with ECOC are 3.27% and 6.61%, respectively. In comparison with these results, the low dimensional subspaces found by gKDR-FEX and gKDR-FEXv maintain the information for classification effectively. SIR-II does not find meaningful features.

The second data set is author identification of Amazon commerce reviews with 10000 dimensional linguistic features. The total number of authors is 50, and 30 reviews were collected for each author; the total size of data is thus 1500. We varied the used number of authors ($L$) to make different levels of difficulty for the tasks. The reduced dimensionality by gKDR-FEX is set to the same as $L$, and the 10-fold CV errors with data projected on the estimated EDR space are evaluated using 1-vs-1 SVM. As comparison, the squared sum of variable-wise Pearson correlations, $\sum_{\ell=1}^{L} \text{Corr}[X^a, Y^\ell]^2$, is also used for choosing explanatory variables ($a = 1, \ldots, 10000$). Such variable selection methods with Pearson correlation are popularly used for very high dimensional data. The variables with top 500 and 2000 correlations are used to make SVM classifiers. As we can see from Table 2, the gKDR-FEX gives much more effective subspaces for regression than the Pearson correlation method, when

the number of authors is large. The creator of the data set has also reported the classification result with a neural network model [15]; for 50 authors, the 10-fold CV error with 2000 selected variables is 19.51%, which is similar to the gKDR-FEX result with only 50 linear features.

# 3 Variable selection with gKDR

In recent years, extensive studies have been done on variable selection with a sparseness penalty ([6, 16, 18, 23–27, 29, 30] among many others). In supervised setting, these studies often consider some specific model for the regression such as least square or logistic regression. While consistency and oracle property have been also established for many methods, the assumption that there is a true parameter in the model may not hold in practice, and thus a strong restriction of the methods. It is then important to consider more flexible ways of variable selection without assuming any parametric model on the regression. The gKDR approach is appealing to this problem, since it realizes conditional independence without strong assumptions for regression or distribution of variables.

Chen et al. [2] recently proposed the Coordinate-Independent Sparse (CIS) method, which is a semi-parametric method for sparse variable selection. In CIS, the linear feature $B^T X$ is assumed with some rows of $B$ zero, but no parametric model is specified for regression. We wish to estimate $B$ so that the zero-rows should be estimated as zeros. This is achieved by imposing the sparseness penalty of the group LASSO [29] in combination with an objective function of linear feature extraction written in the form of eigenproblem such as SIR and PFC [3].

We follow the CIS method for our variable selection with gKDR; since the gKDR is given by the eigenproblem with matrix $\tilde{M}_n$, the CIS method is applied straightforwardly. The significance of our method is that the gKDR formulates the conditional independence of $Y$ and $X$ given $B^T X$, while the existing CIS-based methods in [2] realize only weaker conditions under strong assumptions.

## 3.1 Sparse variable selection with gKDR

Throughout this section, it is assumed that the true probability satisfies Eq. (1) with $B = B_0 = (\mathbf{v}_{01}^T, \dots, \mathbf{v}_{0m}^T)^T$, and with some $1 \leq q \leq m$ the $j$-th row $\mathbf{v}_{0j}$ is non-zero for $j \leq q$ and $\mathbf{v}_{0j} = 0$ for $j \geq q + 1$. The projection matrix is $B = (\mathbf{b}_1, \dots, \mathbf{b}_d) = (\mathbf{v}_1^T, \dots, \mathbf{v}_m^T)^T$, where $\mathbf{b}_i$ is the $i$-th column and $\mathbf{v}_j$ is the $j$-th row. The proposed variable selection method, gKDR-VS, estimates $B$ by

$$\widehat{B}_\lambda = \arg \min_{B:B^T B = I_d} \Big[ -\text{Tr}[B^T \tilde{M}_n B] + \sum_{i=1}^{m} \lambda_i \|\mathbf{v}_i\| \Big], \tag{9}$$

where $\|\mathbf{v}_j\|$ is the Euclidean norm and $\lambda = (\lambda_1, \dots, \lambda_m) \in \mathbb{R}_+^m$ is the regularization coefficients. To optimize Eq. (9), as in [2], we used the local quadratic approximation [6], which is simple and fast. We used the matlab code provided at the homepage of X. Chen.

The choice of $\lambda$ is crucial on the practical performance of sparse variable selection. As a theoretical guarantee, we will show that some asymptotic condition provides model consistency. In practice, as in the Adaptive Lasso [30], it is suitable to consider $\lambda = \lambda(\theta)$ define by
$$\lambda_i = \theta \|\tilde{\mathbf{v}}_i\|^{-r}$$
where $\theta$ and $r$ are positive numbers, and $\tilde{\mathbf{v}}_i$ is the row vector of $\tilde{B}_0$, the solution to gKDR without penalty, i.e., $\tilde{B}_0 = \arg \min_{B^T B = I_d} -\text{Tr}[B^T \tilde{M}_n B]$. We used $r = 1/2$ for all of our experiments.

To choose the parameter $\theta$, a BIC-based method is often used in sparse variable selection [27, 31] with theoretical guarantee of model consistency. We use a BIC-type method for choosing $\theta$ by minimizing

$$\text{BIC}_\theta = -\text{Tr}[\widehat{B}_{\lambda(\theta)}^T \tilde{M}_n \widehat{B}_{\lambda(\theta)}] + C_n \text{df}_\theta \frac{\log n}{n}, \tag{10}$$

where $\text{df}_\theta = d(p - d)$ is the degree of freedom of $\widehat{B}_{\lambda(\theta)}$ with $p$ the number of non-zero rows in $\widehat{B}_{\lambda(\theta)}$, and $C_n$ is a positive number of $O_p(1)$. We used $C_n = \alpha_1 \log \log(m)$ with $\alpha_1$ is the largest eigenvalue of $\tilde{M}_n$. The $\log \log(m)$ factor is used in [27], where increasing number of variables is discussed, and $\alpha_1$ is introduced to adjust the scale of $\text{Tr}[\widehat{B}_\lambda^T \tilde{M}_n \widehat{B}_\lambda]$; we use CV for choosing the hyperparameters (kernel and regularization coefficient), in which the values of $\text{Tr}[\widehat{B}_\lambda^T \tilde{M}_n \widehat{B}_\lambda]$ is not normalized well for different choices.

| | gKDR-VS | CIS-SIR |
|---|---|---|
| (A) $n = 60$ | .94/.99/75 | .89/1.0/65 |
| (A) $n = 120$ | 1.0/1.0/98 | .99/1.0/97 |
| (B) $n = 100$ | .92/.84/63 | .19/.85/1 |
| (B) $n = 200$ | .98/.89/75 | .18/.85/1 |

Table 3: gKDR-VS and CIS-SIR with synthetic data (ratio of non-zeros in $1 \le j \le q$ / ratio of zeros in $q + 1 \le j \le m$ / number of correct models among 100 runs).

| Method | gKDR-VS | | CIS-SIR | | CIS-PFC | |
|---|---|---|---|---|---|---|
| CRIM | 0 | 0 | 0 | 0 | 0 | 0 |
| ZN | 0 | 0 | -0.000 | -0.008 | 0 | 0 |
| INDUS | 0 | 0 | 0 | 0 | 0 | 0 |
| CHAS | 0 | 0 | 0 | 0 | 0 | 0 |
| NOX | 0 | 0 | 0 | 0 | 0 | 0 |
| RM | 0.896 | 0.393 | -1.00 | -1.253 | 1.045 | -1.390 |
| AGE | 0 | 0 | 0.005 | -0.022 | -0.003 | -0.011 |
| DIS | -0.169 | 0.022 | 0 | 0 | 0 | 0 |
| RAD | 0.018 | -0.000 | 0 | 0 | 0 | 0 |
| TAX | 0 | 0 | 0.001 | -0.001 | -0.001 | -0.005 |
| PTRATIO | -0.376 | 0.919 | 0.049 | 0.003 | -0.038 | 0.007 |
| B | 0 | 0 | -0.001 | 0.002 | 0.001 | 0.005 |
| LSTAT | -0.165 | 0.017 | 0.043 | -0.114 | -0.043 | -0.113 |

Table 4: Boston Housing Data: estimated sparse EDR.

## 3.2 Theoretical results on gKDR-VS

This subsection shows the model consistency of the gKDR-VS. All the proofs are shown in Supplements. Let $\alpha_n = \max\{\lambda_j \mid 1 \le j \le q\}$ and $\beta_n = \min\{\lambda_j \mid q + 1 \le j \le m\}$. The eigenvalues of $M = E[M(X)]$ are $\eta_1 \ge \ldots \ge \eta_m \ge 0$. For two $m \times d$ matrices $B_i$ ($i = 1, 2$) with $B_i^T B_i = I_d$, we define $D(B_1, B_2) = \|B_1 B_1^T - B_2 B_2^T\|$, where $\| \cdot \|$ is the operator norm.

**Theorem 3.** *Suppose $\|\tilde{M}_n - M\|_F = O_p(n^{-\tau})$ for some $\tau > 0$. If $n^\tau \alpha_n \to 0$ as $n \to \infty$ and $\eta_q > \eta_{q+1}$, then the estimator $\widehat{B}_\lambda$ in Eq. (9) satisfies $D(\widehat{B}_\lambda, B_0) = O_p(n^{-\tau})$ as $n \to \infty$.*

We saw in Theorem 2 that under some conditions $\tilde{M}_n$ converges to $M$ at the rate $O_p(n^{-\tau})$ with $1/4 \le \tau \le 1/3$. Thus Theorem 3 shows that $\widehat{B}_\lambda$ is also consistent of the same rate.

**Theorem 4.** *In addition to the assumptions in Theorem 3, assume $n^\tau \beta_n \to \infty$ as $n \to \infty$. Then, for all $q + 1 \le j \le m$, $\Pr(\widehat{\mathbf{v}}_j = 0) \to 1$ as $n \to \infty$, where $\widehat{\mathbf{v}}_j$ is the $j$-th row of $\widehat{B}_\lambda$.*

## 3.3 Experiments with gKDR-VS

We first apply the gKDR-VS with $d = 1$ to synthetic data generated by the following two models: (A): $Y = X^1 + X^2 + X^3 + W$ and (B): $Y = (X^1 + X^2 + X^3)^4 W$, where the noise $W$ follows $N(0, 1)$. For (A), $X = (X^1, \ldots, X^{24})$ is generated by $N(0, \Sigma)$ with $\Sigma_{ij} = (1/2)^{|i-j|}$ ($1 \le i, j \le 24$), and for (B) $X = (X^1, \ldots, X^{10})$ by $N(0, 4I_{10})$. Note that (B) includes multiplicative noise, which cannot be handled by many dimension reduction methods. In comparison, the CIS method with SIR is also applied to the same data. The regularization parameter of CIS-SIR is chosen by BIC described in [2]. While both the methods work effectively for (A), only gKDR-VS can handle the multiplicative noise of (C).

The next experiment uses *Boston Housing* data, which has been often used for variable selection. The response $Y$ is the median value of homes in each tract, and thirteen variables are used to explain it. The detail of the variables is described in Supplements, Sec. E. The results of gKDR-VS and CIS-SIR / CIS-PFC with $d = 2$ are shown in Table 4. The variables selected by gKDR-VS are RM, DIS, RAD, PTRATIO and LSTAT, which are slightly different from the CIS methods. In a previous study [1], the four variables RM, TAX, PTRATIO and LSTAT are considered to have major contribution.

## 4 Conclusions

We have proposed a gradient-based kernel approach for dimension reduction in supervised learning. The method is based on the general kernel formulation of conditional independence, and thus has wide applicability without strong restrictions on the model or variables. The linear feature extraction, gKDR-FEX, finds effective features with simple eigendecomposition, even when other conventional methods are not applicable by multiplicative noise or high-dimensionality. The consistency is also guaranteed. We have extended the method to variable selection (gKDR-VS) with a sparseness penalty, and demonstrated its promising performance with synthetic and real world data. The model consistency has been also proved.

**Acknowledgements.** KF has been supported in part by JSPS KAKENHI (B). 22300098.

## Footnotes

[1]Noting $\langle C_{XX}f,f\rangle = E[f(X)^2]$, it is easy to see that $C_{XX}$ is injective, if $k_{\mathcal{X}}$ is a continuous kernel on a topological space $\mathcal{X}$, and $P_X$ is a Borel probability measure such that $P(U) > 0$ for any open set $U$ in $\mathcal{X}$.

# References

[1] L. Breiman and J. Friedman. Estimating optimal transformations for multiple regression and correlation. *J. Amer. Stat. Assoc.*, 80:580–598, 1985.

[2] X. Chen, C. Zou, and R. Dennis Cook. Coordinate-independent sparse sufficient dimension reduction and variable selection. *Ann. Stat.*, 38(6):3696–3723, 2010.

[3] R. Dennis Cook and L. Forzani. Principal fitted components for dimension reduction in regression. *Statistical Science*, 23(4):485–501, 2008.

[4] R. Dennis Cook and S. Weisberg. Discussion of Li (1991). *J. Amer. Stat. Assoc.*, 86:328–332, 1991.

[5] J. Fan and I. Gijbels. *Local Polynomial Modelling and its Applications*. Chapman and Hall, 1996.

[6] J. Fan and R. Li. Variable selection via nonconcave penalized likelihood and its oracle properties. *J. Amer. Stat. Assoc.*, 96(456):1348–1360, 2001.

[7] A. Frank and A. Asuncion. UCI machine learning repository, [http://archive.ics.uci.edu/ml]. Irvine, CA: University of California, School of Information and Computer Science. 2010.

[8] K. Fukumizu, F.R. Bach, and M.I. Jordan. Dimensionality reduction for supervised learning with reproducing kernel Hilbert spaces. *JMLR*, 5:73–99, 2004.

[9] K. Fukumizu, F.R. Bach, and M.I. Jordan. Kernel dimension reduction in regression. *Ann. Stat.*, 37(4):1871–1905, 2009.

[10] A. Gretton, K. Fukumizu, C.H. Teo, L. Song, B. Schölkopf, and Alex Smola. A kernel statistical test of independence. In *Advances in NIPS 20*, pages 585–592. 2008.

[11] M. Hristache, A. Juditsky, J. Polzehl, and V. Spokoiny. Structure adaptive approach for dimension reduction. *Ann. Stat.*, 29(6):1537–1566, 2001.

[12] B. Li, H. Zha, and F. Chiaromonte. Contour regression: A general approach to dimension reduction. *Ann. Stat.*, 33(4):1580–1616, 2005.

[13] K.-C. Li. Sliced inverse regression for dimension reduction (with discussion). *J. Amer. Stat. Assoc.*, 86:316–342, 1991.

[14] K.-C. Li. On principal Hessian directions for data visualization and dimension reduction: Another application of Stein's lemma. *J. Amer. Stat. Assoc.*, 87:1025–1039, 1992.

[15] S. Liu, Z. Liu, J. Sun, and L. Liu. Application of synergetic neural network in online writeprint identification. *Intern. J. Digital Content Technology and its Applications*, 5(3):126–135, 2011.

[16] L. Meier, S. Van De Geer, and P. Bühlmann. The group lasso for logistic regression. *J. Royal Stat. Soc.: Ser. B*, 70(1):53–71, 2008.

[17] A.M. Samarov. Exploring regression structure using nonparametric functional estimation. *J. Amer. Stat. Assoc.*, 88(423):836–847, 1993.

[18] S. K. Shevade and S. S. Keerthi. A simple and efficient algorithm for gene selection using sparse logistic regression. *Bioinformatics*, 19(17):2246–2253, 2003.

[19] L. Song, J. Huang, A. Smola, and K. Fukumizu. Hilbert space embeddings of conditional distributions with applications to dynamical systems. In *Proc. ICML2009*, pages 961–968. 2009.

[20] B. K. Sriperumbudur, A. Gretton, K. Fukumizu, B. Schölkopf, and G.R.G. Lanckriet. Hilbert space embeddings and metrics on probability measures. *JMLR*, 11:1517–1561, 2010.

[21] I. Steinwart and A. Christmann. *Support Vector Machines*. Springer, 2008.

[22] G.W. Stewart and J.-Q. Sun. *Matrix Perturbation Theory*. Academic Press, 1990.

[23] R. Tibshirani. Regression shrinkage and selection via the lasso. *J. Royal Stat. Soc.: Ser. B*, 58(1):pp. 267–288, 1996.

[24] H. Wang and C. Leng. Unified lasso estimation by least squares approximation. *J. Amer. Stat. Assoc.*, 102 (479):1039–1048, 2007.

[25] H. Wang, G. Li, and C.-L. Tsai. Regression coefficient and autoregressive order shrinkage and selection via the lasso. *J. Royal Stat. Soc.: Ser. B*, 69(1):63–78, 2007.

[26] H. Wang, G. Li, and C.-L. Tsai. On the consistency of SCAD tunign parameter selector. *Biometrika*, 94: 553–558, 2007.

[27] H. Wang, B. Li, and C. Leng. Shrinkage tuning parameter selection with a diverging number of parameters. *J. Royal Stat. Soc.: Ser. B*, 71(3):671–683, 2009.

[28] M. Wang, F. Sha, and M. Jordan. Unsupervised kernel dimension reduction. *NIPS 23*, pages 2379–2387. 2010.

[29] M. Yuan and Y. Lin. Model selection and estimation in regression with grouped variables. *J. Royal Stat. Soc.: Ser. B*, 68(1):49–67, 2006.

[30] H. Zou. The adaptive lasso and its oracle properties. *J. Amer. Stat. Assoc.*, 101:1418–1429, 2006.

[31] C. Zou and X. Chen. On the consistency of coordinate-independent sparse estimation with BIC. *J. Multivariate Analysis*, 112:248–255, 2012.

